# Learning to use Working Memory in Partially Observable Environments through Dopaminergic Reinforcement

**Michael T. Todd, Yael Niv, Jonathan D. Cohen**
Department of Psychology & Princeton Neuroscience Institute
Princeton University, Princeton, NJ 08544
{*mttodd,yael,jdc*}*@princeton.edu*

## Abstract

Working memory is a central topic of cognitive neuroscience because it is critical for solving real-world problems in which information from multiple temporally distant sources must be combined to generate appropriate behavior. However, an often neglected fact is that learning to use working memory effectively is itself a difficult problem. The Gating framework [1-4] is a collection of psychological models that show how dopamine can train the basal ganglia and prefrontal cortex to form useful working memory representations in certain types of problems. We unite Gating with machine learning theory concerning the general problem of memory-based optimal control [5-6]. We present a normative model that learns, by online temporal difference methods, to use working memory to maximize discounted future reward in partially observable settings. The model successfully solves a benchmark working memory problem, and exhibits limitations similar to those observed in humans. Our purpose is to introduce a concise, normative definition of high level cognitive concepts such as working memory and cognitive control in terms of maximizing discounted future rewards.

## 1      Introduction

Working memory is loosely defined in cognitive neuroscience as information that is (1) internally maintained on a temporary or short term basis, and (2) required for tasks in which immediate observations cannot be mapped to correct actions. It is widely assumed that prefrontal cortex (PFC) plays a role in maintaining and updating working memory. However, relatively little is known about how PFC develops useful working memory representations for a new task. Furthermore, current work focuses on describing the structure and limitations of working memory, but does not ask why, or in what general class of tasks, is it necessary. Borrowing from the theory of optimal control in partially observable Markov decision problems (POMDPs), we frame the psychological concept of working memory as an internal state representation, developed and employed to maximize future reward in partially observable environments. We combine computational insights from POMDPs and neurobiologically plausible models from cognitive neuroscience to suggest a simple reinforcement learning (RL) model of working memory function that can be implemented through dopaminergic training of the basal ganglia and PFC.

The Gating framework is a series of cognitive neuroscience models developed to explain how dopaminergic RL signals can shape useful working memory representations [1-4]. Computationally this framework models working memory as a collection of past observations, each of which can occasionally be replaced with the current observation, and addresses the problem of learning *when* to update each memory element versus maintaining it. In the original Gating model [1-2] the PFC contained a unitary working memory

representation that was updated whenever a phasic dopamine (DA) burst occurred (e.g., due to unexpected reward or novelty). That model was the first to connect working memory and RL via the temporal difference (TD) model of DA firing [7-8], and thus to suggest how working memory might serve a normative purpose. However, that model had limited computational flexibility due to the unitary nature of the working memory (i.e., a single-observation memory controlled by a scalar DA signal). More recent work [3-4] has partially repositioned the Gating framework within the Actor/Critic model of mesostriatal RL [9-10], positing memory updating as but another cortical action controlled by the dorsal striatal "actor." This architecture increased computational flexibility by introducing multiple working memory elements, corresponding to multiple corticostriatal loops, that could be quasi-independently updated. However, that model combined a number of components (including supervised and unsupervised learning, and complex neural network dynamics), making it difficult to understand the relationship between simple RL mechanisms and working memory function. Moreover, because the model used the Rescorla-Wagner-like PVLV algorithm [4] rather than TD [7-8] as the model of phasic DA bursts, the model's behavior and working memory representations were not directly shaped by standard normative criteria for RL models (i.e., discounted future reward or reward per unit time).

We present a new Gating model, synthesizing the mesostriatal Actor/Critic architecture of [4] with a normative POMDP framework, and reducing the Gating model to a four-parameter, pure RL model in the process. This produces a model very similar to previous machine learning work on "model-free" approximate POMDP solvers [5,6], which attempt to form good solutions without explicit knowledge of the environment's structure or dynamics. That is, we model working memory as a discrete memory system (a collection of recent observations) rather than a continuous "belief state" (an inferred probability distribution over hidden states). In some environments this may permit only an approximate solution. However, the strength of such a system is that it requires very little prior knowledge, and is thus potentially useful for animals, who must learn effective behavior and memory-management policies in completely novel environments (i.e., in the absence of a "world model"). Therefore, we retain the computational flexibility of the more recent Gating models [3-4], while re-establishing the goal of defining working memory in normative terms [1-2].

To illustrate the strengths and limitations of the model, we apply it to two representative working-memory tasks. The first is the 12-AX task proposed as a Gating benchmark in [4]. Contrary to previous claims that TD learning is not sufficient to solve this task, we show that with an eligibility trace (i.e., TD($\lambda$) with $0 < \lambda < 1$), the model can achieve optimal behavior. The second task highlights important limitations of the model. Since our model is a POMDP solver and POMDPs are, in general, intractable (i.e., solution algorithms require an infeasible number of computations), it is clear that our model must ultimately fail to achieve optimal performance as environments increase even to moderate complexity. However, human working memory also exhibits sharp limitations. We apply our model to an implicit artificial grammar learning task [11] and show that it indeed fails in ways reminiscent of human performance. Moreover, simulating this task with increased working memory capacity reveals diminishing returns as capacity increases beyond a small number, suggesting that the "magic number" limited working memory capacity found in humans [12] might in fact be optimal from a learning standpoint.

## 2    Model Architecture

As with working memory tasks, a POMDP does not admit an optimal behavior policy based only on the current observation. Instead, the optimal policy generally depends on some combination of memory as well as the current observation. Although the type of memory required varies across POMDPs, in certain cases a *finite* memory system is a sufficient basis for an optimal policy. Peshkin, Meuleau, and Kaelbling [6] used an external finite memory device (e.g., a shopping list) to improve the performance of RL in a model-free POMDP setting. Their model's "state" variable consisted of the current observation augmented by the memory device. An augmented action space, consisting of both memory actions and motor actions, allowed the model to learn effective memory-management and motor policies simultaneously. We integrate this approach with the Gating model, altering the semantics so that the external memory device becomes internal working memory (presumed

| | |
|---|---|
| Choose motor action, $a_t$, and gating action, $g_t$, for current state, $s_t$ according to softmax over motor and gating action preferences, $u$ and $v$, respectively. | $a_t \leftarrow \text{Softmax}(u; s_t)$ <br> $g_t \leftarrow \text{Softmax}(v; s_t)$ |
| Update motor and gating action eligibility traces, $e^M$ and $e^G$, respectively. (Update shown for motor action eligibility trace. Gating action trace is analogous.) | $e^M(s,a) \leftarrow \begin{cases} 1 - \Pr(a\|s), & s = s_t, a = a_t \\ -\Pr(a\|s), & s = s_t, a \neq a_t \quad \forall s,a \\ \gamma\lambda e^M(s,a), & s \neq s_t \end{cases}$ |
| Update (hidden) environment state, $\sigma$, with motor action. Get next reward, $r$, and observation, $o$. | $\sigma_{t+1} \leftarrow \text{Environment}(a_t, \sigma_t)$ <br> $r, o \leftarrow \text{Environment}(\sigma_{t+1})$ |
| Update internal state based on previous state, gating action, and new observation | $s_{t+1} \leftarrow o, g_t, s_t$ |
| Compute state-value prediction error, $\delta_t$, based on critic's state-value approximation, $V(s)$ | $\delta_t \leftarrow r + \gamma V(s_{t+1}) - V(s_t)$ |
| Update state-value eligibility traces, $e^V$. | $e^V(s) = \begin{cases} \gamma\lambda e^V(s) + 1, & s = s_t \\ \gamma\lambda e^V(s), & s \neq s_t \end{cases}, \ \forall s$ |
| Update state-values | $V(s) = V(s) + \alpha\delta_t e^V(s), \ \forall s$ |
| Update motor action preferences | $u(s,a) = u(s,a) + \alpha\delta_t e^M(s,a), \qquad \forall s,a$ |
| Update gating action preferences | $v(s,g) = v(s,g) + \alpha\delta_t e^G(s,g), \qquad \forall s,g$ |
| Next trial… | $s_t \leftarrow s_{t+1}$ |

Table 1 Pseudocode of one trial of the model, based on the Actor/Critic architecture with eligibility traces. Following [13], we substitute the critic's state-value prediction error for Williams's $(r - b)$ term [14]. We describe here a single gating actor, but it is straightforward to generalize to an array of independent gating actors as we use in our simulations. $\gamma$ = discount rate; $\lambda$ = eligibility trace decay rate; $\alpha$ =learning rate. In all simulations, $\gamma = 0.94$, $\alpha = 0.1$.

to be supported in PFC), and altering the Gating model so that the role of working memory is explicitly to support optimal behavior (in terms of discounted future reward) in a POMDP.

Like [6], the key difference between our model and standard RL methods is that our state variable includes controlled memory elements (i.e., working memory), which augment the current observation. The action space is similarly augmented to include memory or gating actions, and the model learns by trial-and-error how to update its working memory (to resolve hidden states when such resolution leads to greater rewards) as well as its motor policy. The task for our model then, is to learn a working memory policy such that the current *internal state* (i.e., memory and current observation) admits an optimal behavioral policy.

Our model (Table 1) consists of a critic, a *motor actor,* and several *gating actors*. As in the standard Actor/Critic architecture, the critic learns to evaluate (internal) states and, based on the ongoing temporal difference of these values, generates at each time step a prediction error (PE) signal (thought to correspond to phasic bursts and dips in DA [8]). The PE is used to train the critic's state values and the policies of the actors. The motor actor also fulfills the usual role, choosing actions to send to the environment based on its policy and the current internal state. Finally, gating actors correspond one-to-one with each memory element. At each time point, each gating actor independently chooses (via a policy based on the internal state) whether to (1) *maintain* its element's memory for another time step, or (2) *replace* (update) its element's memory with the current observation.

To remain aligned with the Actor/Critic online learning framework of mesostriatal RL [9-10], learning in our model is based on REINFORCE [14] modified for expected discounted future reward [13], rather than the Monte-Carlo policy learning algorithm in [6] (which is more suitable for offline, episodic learning). Furthermore, because it has been shown that eligibility traces are particularly useful when applying TD to POMDPs (e.g., [15-16]), we used TD($\lambda$), taking the characteristic eligibilities of the REINFORCE algorithm [14] as the impulse function for a replacing eligibility trace [17]. For simplicity of exposition and interpretation, we used tabular policy and state-value representations throughout.

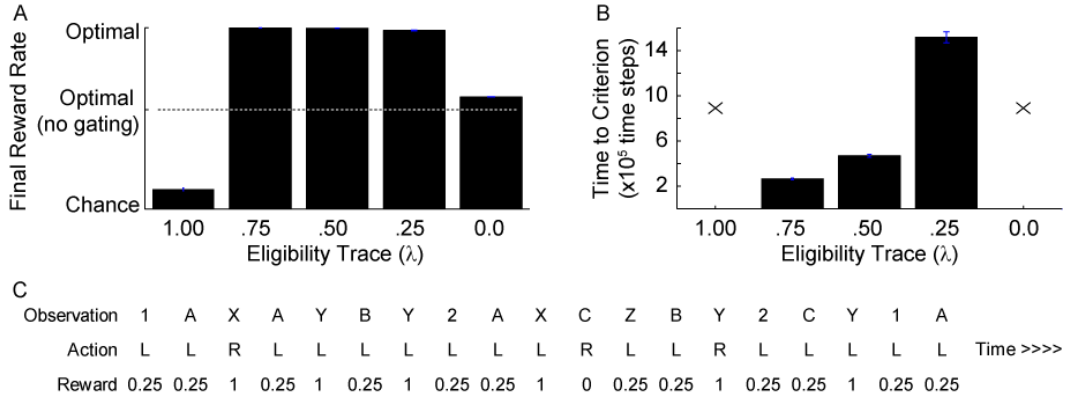

Figure 1 12-AX: Average performance over 40 training runs, each consisting of $2 \times 10^7$ timesteps. (A) As indicated by reward rate over the last $10^5$ time steps, the model learns an optimal policy when the eligibility trace parameter, $\lambda$, is between zero and one. (B) The time required for the model to reach 300 consecutive correct trials increases rapidly as $\lambda$ decreases. (C) Sample sequence of the 12-AX task.

# 3    Benchmark Performance and Psychological Data

We now describe the model's performance on the 12-AX task proposed as a benchmark for Gating models [4]. We then turn to a comparison of the model's behavior against actual psychological data.

## 3.1    12-AX Performance

The 12-AX task was used in [4] to illustrate the problem of learning a task in which correct behavior depends on multiple previous observations. In the task (Figure 1C), subjects are presented with a sequence of observations drawn from the set {1, 2, A, B, C, X, Y, Z}. They gain rewards by responding L or R according to the following rules: Respond R if (1) the current observation is an X, the last observation from the set {A, B, C} was an A, and the last observation from the set {1, 2} was a 1; or (2) the current observation is a Y, the last observation from the set {A, B, C} was a B, and the last observation from the set {1, 2} was a 2. Respond L otherwise. In our implementation, reward is 1 for correct responses when the current observation is X or Y, 0.25 for all other correct responses, and 0 for incorrect responses.

We modeled this task using two memory elements, the minimum theoretically necessary for optimal performance. The results (Figure 1A,B) show that our TD($\lambda$) Gating model can indeed achieve optimal 12-AX performance. The results also demonstrate the reliance of the model on the eligibility trace parameter, $\lambda$, with best performance at high intermediate values of $\lambda$. When $\lambda = 0$, the model finds a suboptimal policy that is only slightly better than the optimal policy for a model without working memory. With $\lambda = 1$ performance is even worse, as can be expected for an online policy improvement method with non-decaying traces (a point of comparison with [6] to which we will return in the Discussion). These results are consistent with previous work showing that TD(0) performs poorly in partially observable (non-Markovian) settings [15], whereas TD($\lambda$) (without memory) with $\lambda \approx 0.9$ performs best [16]. Indeed, early in training, as our model learns to convert a POMDP to an MDP via its working memory, the internal state dynamics are not Markovian, and thus an eligibility trace is necessary.

## 3.2    Psychological data

We are the first to interpret the Gating framework (and the use of working memory) as an attempt to solve POMDPs. This brings a large body of theoretical work to bear on the properties of Gating models. Importantly, it implies that, as task complexity increases, both the Gating model and humans must fail to find optimal solutions in reasonable time frames

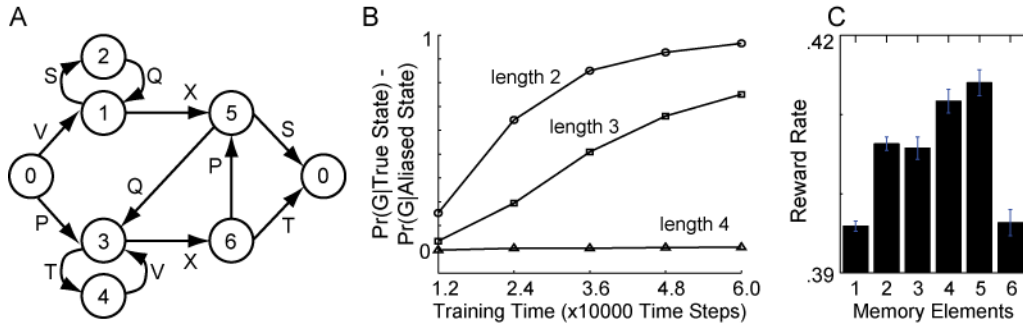

Figure 2 (A) Artificial grammar from [11]. Starting from node 0, the grammar generates a continuing sequence of observations. All nodes with two transitions (edges) make either transition with p=0.5. Edge labels mark grammatical observations. At each transition, the grammatical observation is replaced with a random, ungrammatical, observation with p=0.15. The task is to predict the next observation at each time point. (B) The model shows a gradual increase in sensitivity to sequences of length 2 and 3, but not length 4, replicating the human data. Sensitivity is measured as probability of choosing grammatical action for the true state, minus probability of choosing grammatical action for the aliased state; 0 indicates complete aliasing, 1 complete resolution. (C) Model performance (reward rate) averaged over training runs with variable numbers of time steps shows diminishing returns as the number of memory elements increases.

due to the generally intractable nature of POMDPs. Given this inescapable conclusion, it is interesting to compare model failures to corresponding human failures: a pattern of failures matching human data would provide support for our model. In this subsection we describe a simulation of artificial grammar learning [11], and then offer an account of the pervasive "magic number" observations concerning limits of working memory capacity (e.g., [12]).

In artificial grammar learning, subjects see a seemingly random sequence of observations, and are instructed to mimic each observation as quickly as possible (or to predict the next observation) with a corresponding action. Unknown to the subjects, the observation sequence is generated by a stochastic process called a "grammar" (Figure 2A). Artificial grammar tasks constitute POMDPs: the (recent) observation history can predict the next observation better than the current observation alone, so optimal performance requires subjects to remember information distilled from the history. Although subjects typically report no knowledge of the underlying structure, after training their reaction times (RTs) reveal implicit structural knowledge. Specifically, RTs become significantly faster for "grammatical" as compared to "ungrammatical" observations (see Figure 2).

Cleeremans and McClelland [11] examined the limits of subjects' capacity to detect grammar structure. The grammar they used is shown in Figure 2A. They found that, although subjects grew increasingly sensitive to sequences of length two and three throughout training, (as measured by transient RT increases following ungrammatical observations), they remained insensitive, even after 60,000 time steps of training, to sequences of length four. This presumably reflected a failure of subjects' implicit working memory learning mechanisms, and was confirmed in a second experiment [11]. We replicated these results, as shown in Figure 2B. To simulate the task, we gave the model two memory elements (results were no different with three elements), and reward 1 for each correct prediction. We tested the model's ability to resolve states based on previous observations by contrasting its behavior across pairs of observation sequences that differed only in the first observation. State resolution based on sequences of length two, three, and four were represented by VS versus XS (leading to predictions Q vs. V/P, respectively), SQX versus XQX (S/Q vs. P/T), and XTVX versus PTVX (S/Q vs. P/T), respectively.

In this task, optimal use of information from sequences of length four or more proved impossible for the model and, apparently, for humans. To understand intuitively this limitation, consider a problem of two hidden states, 1 and 2, with optimal actions L and R, respectively. The states are preceded by identical observation sequences of length    . However, at     + *1* time steps in the past, observation A precedes state 1, whereas

observation B precedes state 2. The probability that A/B are held in memory for the required $+1$ time steps decreases geometrically with , thus the probability of resolving states 1 and 2 decreases geometrically. Because the agent cannot resolve state 1 from state 2, it can never learn the appropriate 1-L, 2-R action preferences *even if it explores those actions*, a more insidious problem than an RL agent faces in a fully observable setting. As a result, the model can't reinforce optimal gating policies, eventually learning an internal state space and dynamics that fail to reflect the true environment. The problem is that credit assignment (i.e., learning a mapping from working memory to actions) is only useful inasmuch as the internal state corresponds to the true hidden state of the POMDP, leading to a "chicken-and-egg" problem.

Given the preceding argument, one obvious modification that might lead to improved performance is to increase the number of memory elements. As the number of memory elements increases, the probability that the model remembers observation A for the required amount of time approaches one. However, this strategy introduces the curse of dimensionality due to the rapidly increasing size of the internal state space.

This intuitive analysis suggests a normative explanation for the famous "magic number" limitation observed in human working memory capacity, thought to be about four independent elements (e.g., [12]). We demonstrate this idea by again simulating the artificial grammar task, this time averaging performance over a range of training times (1 to 10 million time steps) to capture the idea that humans may practice novel tasks for a typical, but variable, amount of time. Indeed the averaged results show diminishing returns of increasing memory elements (Figure 2C). This simulation used tabular (rather than more neurally plausible) representations and a highly simplified model, so the exact number of policy parameters and state values to be estimated, time steps, and working memory elements is somewhat arbitrary in relation to human learning. Still, the model's qualitative behavior (evidenced by the shape of the resulting curve and the order of magnitude of the optimal number of working memory elements) is surprisingly reminiscent of human behavior. Based on this we suggest that the limitation on working memory capacity may be due to a limitation on learning rather than on storage: it may be impractical to learn to utilize more than a very small number (i.e., smaller than 10) of independent working memory elements, due to the curse of dimensionality.

# 4    Discussion

We have presented a psychological model that suggests that dopaminergic PE signals can implicitly shape working memory representations in PFC. Our model synthesizes recent advances in the Gating literature [4] with normative RL theory regarding model-free, finite memory solutions to POMDPs [6]. We showed that the model learns to behave optimally in the benchmark 12-AX task. We also related the model's computational limitations to known limitations of human working memory [11-12].

## 4.1    Relation to other theoretical work

Other recent work in neural RL has argued that the brain applies memory-based POMDP solution mechanisms to the real-world problems faced by animals [17-20]. That work primarily considers model-*based* mechanisms, in which the temporary memory is a continuous belief state, and assumes that a function of cerebral cortex is to learn the required world model, and specifically that PFC should represent temporary goal- or policy-related information necessary for optimal POMDP behavior. The model that we present here is related to that line of thinking, demonstrating a model-free, rather than model-based, mechanism for learning to store policy-related information in PFC. Different learning systems may form different types of working memory representations. Future work may investigate the relationship between implicit learning (as in this Gating model) and model-free POMDP solutions, versus other kinds of learning and model-based POMDP solutions. Irrespective of the POMDP framework, other work has assumed that there exists a gating policy that controls task-relevant working memory updating in PFC (e.g., [21]). The present work further develops a model of how this policy can be learned.

It is interesting to compare our model to previous work on model-free POMDP solutions.

McCallum first emphasized the importance of learning *utile distinctions* [5], or learning to resolve two hidden states only if they have different optimal actions. This is an emphasis that our model shares, at least in spirit. Humans must of course be extremely flexible in their behavior. Therefore there is an inherent tension between the need to focus cognitive resources on learning the immediate task, and the need to form a basis of general task knowledge [3]. It would be interesting for future work to explore how closely the working memory representations learned by our model align to McCallum's utile (and less generalizable) distinctions as opposed to more generalizable representations of the underlying hidden structure of the world, or whether our model could be modified to incorporate a mixture of both kinds of knowledge, depending on some exploration/exploitation parameter.

Our model most closely follows the Gating model described in [4], and the theoretical model described in [6]. Our model is clearly more abstract and less biologically detailed than [4]. However, our intent was to ask whether the important insights and capabilities of that model could be captured using a four-parameter, pure RL model with a clear normative basis. Accordingly, we have shown that such a model is comparably equipped to simulate a range of psychological phenomena. Our model also makes equally testable (albeit different) predictions about the neural DA signal. Relative to [6], our model places biological and psychological concerns at the forefront, eliminating the episodic memory requirements of the Monte-Carlo algorithm. It is perhaps interesting, vis á vis [6], that our model performed so poorly when $\quad = 1$, as this produces a nearly Monte-Carlo scheme. The difference was likely due to our model's online learning (i.e., we updated the policy at each time step rather than at the ends of episodes), which invalidates the Monte-Carlo approach. Thus it might be said that our model is a uniquely psychological variant of that previous architecture.

## 4.2    Implications for Working Memory and Cognitive Control

Subjects in cognitive control experiments typically face situations in which correct behavior is indeterminate given only the immediate observation. Working memory is often thought of as the repository of temporary information that augments the immediate observation to permit correct behavior, sometimes called goals, context, task set, or decision categories. These concepts are difficult to define. Here we have proposed a formal theoretical definition for the cognitive control and working memory constructs. Due to the importance of temporally distant goals and of information that is not immediately observable, the canonical cognitive control environment is well captured by a POMDP. Working memory is then the temporary information, defined and updated by a memory control policy, that the animal uses to solve these POMDPs. Model-*based* research might identify working memory with continuous belief states, whereas our model-*free* framework identifies working memory with a discrete collection of recent observations. These may correspond to the products of different learning systems, but the outcome is the same in either case: cognitive control is defined as an animal's memory-based POMDP solver, and working memory is defined as the information, derived from recent history, that the solver requires.

## 4.3    Psychological and neural validity

Although the intractability of solving a POMDP means that all models such as the one we present here must ultimately fail to find an optimal solution in a practical amount of time (if at all), the particular manifestation of computational limitations in our model aligns qualitatively with that observed in humans. Working memory, the psychological construct that the Gating model addresses, is famously limited (see [12] for a review). Beyond canonical working memory capacity limitations, other work has shown subtler limitations arising in learning contexts (e.g., [11]). The results that we presented here are promising, but it remains for future work to more fully explore the relation between the failures exhibited by this model and those exhibited by humans.

In conclusion, we have shown that the Gating framework provides a connection between high level cognitive concepts such as working memory and cognitive control, systems neuroscience, and current neural RL theory. The framework's trial-and-error method for solving POMDPs gives rise to particular limitations that are reminiscent of observed

psychological limits. It remains for future work to further investigate the model's ability to capture a range of specific psychological and neural phenomena. Our hope is that this link between working memory and POMDPs will be fruitful in generating new insights, and suggesting further experimental and theoretical work.

**Acknowledgments**

We thank Peter Dayan, Randy O'Reilly, and Michael Frank for productive discussions, and three anonymous reviewers for helpful comments. This work was supported by NIH grant 5R01MH052864 (MT & JDC) and a Human Frontiers Science Program Fellowship (YN)

**References**

[1] Braver, T. S., & Cohen, J. D. (1999). Dopamine, cognitive control, and schizophrenia: The gating model. In J. A. Reggia, E. Ruppin, & D. Glanzman (Eds.), *Progress in Brain Research* (pp. 327-349). Amsterdam, North-Holland: Elsevier Science.

[2] Braver, T. S., & Cohen, J. D. (2000). On the Control of Control: The Role of Dopamine in Regulating Prefrontal Function and Working Memory. In S. Monsell, & J. S. Driver (Eds.), *Control of Cognitive Processes: Attention and Performance XVIII* (pp. 713-737). Cambridge, MA: MIT Press.

[3] Rougier, A., Noelle, D., Braver, T., Cohen, J., & O'Reilly, R. (2005). Prefrontal Cortex and Flexible Cognitive Control: Rules Without Symbols. *Proceedings of the National Academy of Sciences , 102* (20), 7338-7343.

[4] O'Reilly, R. C., & Frank, M. J. (2006). Making Working Memory Work: A Computational Model of Learning in the Prefrontal Cortex and Basal Ganglia. *Neural Computation , 18*, 283-328.

[5] McCallum, A. (1995). Instance-Based Utile Distinctions for Reinforcement Learning with Hidden State. *International Conference on Machine Learning*, (pp. 387-395).

[6] Peshkin, L., Meuleau, N., & Kaelbling, L. (1999). Learning Policies with External Memory. *Sixteenth International Conference on Machine Learning*, (pp. 307-314).

[7] Montague, P. R., Dayan, P., & Sejnowski, T. J. (1996). A Framework for Mesencephalic Dopamine Systems Based on Predictive Hebbian Learning. *The Journal of Neuroscience , 16* (5), 1936-1947.

[8] Schultz, W., Dayan, P., & Montague, P. R. (1997). A Neural Substrate of Prediction and Reward. *Science , 275*, 1593-1599.

[9] Houk, J., Adams, J., & Barto, A. (1995). A Model of how the Basal Ganglia Generate and use Neural Signals that Predict Reinforcement. In J. Houk, J. Davis, & D. Beiser, *Models of Information Processing in the Basal Ganglia.* MIT Press.

[10] Joel, D., Niv, Y., & Ruppin, E. (2002). Actor-critic Models of the Basal Ganglia: New Anatomical and Computational Perspectives. *Neural Networks , 15*, 535-547.

[11] Cleeremans, A., & McClelland, J. (1991). Learning the Structure of Event Sequences. *Journal of Experimental Psychology: General , 120* (3), 235-253.

[12] Cowan, N. (2000). The Magical Number 4 in Short-term Memory: A Reconsideration of Mental Storage Capacity. *Behavioral and Brain Sciences , 24*, 87-114.

[13] Dayan, P., & Abbott, L. (2001). *Theoretical Neuroscience.* Cambridge, MA: MIT Press.

[14] Williams, R. (1992). Simple Statistical Gradient-Following Algorithms for Connectionist Reinforcement Learning. *Machine Learning , 8*, 229-256.

[15] Singh, S., Jaakkola, T., & Jordan, M. I. (1994). Learning Without State-Estimation in Partially Observable Markovian Decision Processes. *Eleventh International Conference on Machine Learning*, (pp. 284-292).

[16] Loch, J., & Singh, S. (1998). Using Eligibility Traces to Find the Best Memoryless Policy in Partially Observable Markov Decision Processes. *Fifteenth International Conference on Machine Learning*, (pp. 323-331).

[17] Sutton, R., & Barto, A. (1998). *Reinforcement Learning: An Introduction.* Cambridge, MA: The MIT Press.

[18] Daw, N., Courville, A., & Touretzky, D. (2006). Representation and Timing in Theories of the Dopamine System. *Neural Computation , 18*, 1637-1677.

[19] Samejima, K., & Doya, K. (2007). Multiple Representations of Belief States and Action Values in Corticobasal Ganglia Loops. *Annals of the New York Academy of Sciences* , 213-228.

[20] Yoshida, W., & Ishii, S. (2006). Resolution of Uncertainty in Prefrontal Cortex. *Neuron , 50*, 781-789.

[21] Dayan, P. (2007). Bilinearity, Rules, and Prefrontal Cortex. *Frontiers in Computational Neuroscience , 1*, 1-14.
